# Fast Embedding of
# Sparse Music Similarity Graphs

**John C. Platt**
Microsoft Research
1 Microsoft Way
Redmond, WA 98052 USA
jplatt@microsoft.com

## Abstract

This paper applies fast sparse multidimensional scaling (MDS) to a large graph of music similarity, with 267K vertices that represent artists, albums, and tracks; and 3.22M edges that represent similarity between those entities. Once vertices are assigned locations in a Euclidean space, the locations can be used to browse music and to generate playlists.

MDS on very large sparse graphs can be effectively performed by a family of algorithms called Rectangular Dijsktra (RD) MDS algorithms. These RD algorithms operate on a dense rectangular slice of the distance matrix, created by calling Dijsktra a constant number of times. Two RD algorithms are compared: Landmark MDS, which uses the Nyström approximation to perform MDS; and a new algorithm called Fast Sparse Embedding, which uses FastMap. These algorithms compare favorably to Laplacian Eigenmaps, both in terms of speed and embedding quality.

## 1   Introduction

This paper examines a general problem: given a sparse graph of similarities between a set of objects, quickly assign each object a location in a low-dimensional Euclidean space. This general problem can arise in several different applications: the paper addresses a specific application to music similarity.

In the case of music similarity, a set of musical entities (i.e., artists, albums, tracks) must be placed in a low-dimensional space. Human editors have already supplied a graph of similarities, e.g., artist A is similar to artist B. There are three good reasons to embed a musical similarity graph:

1. *Visualization* — If a user's musical collection is placed in two dimensions, it can be easily visualized on a display. This visualization can aid musical browsing.

2. *Interpolation* — Given a graph of similarities, it is simple to find music that "sounds like" other music. However, once music is embedded in a low-dimensional space, new user interfaces are enabled. For example, a user can specify a playlist by starting at song A and ending at song B, with the songs in the playlist smoothly interpolating between A and B.

3. *Compression* — In order to estimate "sounds like" directly from a graph of music similarities, the user must have access to the graph of all known music. However, once all of the musical entities are embedded, the coordinates for the music in a user's collection can be shipped down to the user's computer. These coordinates are much smaller than the entire graph.

It is important to have algorithms that exploit the sparseness of similarity graphs because large-scale databases of similarities are very often sparse. Human editors cannot create a dense $N \times N$ matrix of music similarity for large values of $N$. The best editors can do is identify similar artists, albums, and tracks. Furthermore, humans are poor at accurately estimating large distances between entities (e.g., which is farther away from The Beatles: Enya or Duke Ellington?)

Hence, there is a definite need for an scalable embedding algorithm that can handle a sparse graph of similarities, generalizing to similarities not seen in the training set.

## 1.1 Structure of Paper

The paper describes three existing approaches to the sparse embedding problem in section 2 and section 3 describes a new algorithm for solving the problem. Section 4.1 verifies that the new algorithm does not get stuck in local minima and section 4.2 goes into further detail on the application of embedding musical similarity into a low-dimensional Euclidean space.

## 2 Methods for Sparse Embedding

Multidimensional scaling (MDS) [4] is an established branch of statistics that deals with embedding objects in a low-dimensional Euclidean space based on a matrix of similarities. More specifically, MDS algorithms take a matrix of *dis*similarities $\delta_{rs}$ and find vectors $\vec{x}_r$ whose inter-vector distances $d_{rs}$ are well matched to $\delta_{rs}$. A common flexible algorithm is called ALSCAL [13], which encourages the inter-vector distances to be near some ideal values:

$$\min_{\vec{x}_r} \sum_{rs} (d_{rs}^2 - \hat{d}_{rs}^2)^2, \tag{1}$$

where $\hat{d}$ are derived from the dissimilarities $\delta_{rs}$, typically through a linear relationship.

There are three existing approaches for applying MDS to large sparse dissimilarity matrices:

1. *Apply an MDS algorithm to the sparse graph directly.*

Not all MDS algorithms require a dense matrix $\delta_{rs}$. For example, ALSCAL can operate on a sparse matrix by ignoring missing terms in its cost function (1). However, as shown in section 4.1, ALSCAL cannot reconstruct the position of known data points given a sparse matrix of dissimilarities.

2. *Use a graph algorithm to generate a full matrix of dissimilarities.*

The Isomap algorithm [14] finds an embedding of a sparse set of dissimilarities into a low-dimensional Euclidean space. Isomap first applies Floyd's shortest path algorithm [9] to find the shortest distance between any two points in the graph, and then uses these $N \times N$ distances as input to a full MDS algorithm. Once in the low-dimensional space, data can easily be interpolated or extrapolated. Note that the systems in [14] have $N = 1000$.

For generalizing musical artist similarity, [7] also computes an $N \times N$ matrix of distances between all artists in a set, based on the shortest distance through a graph. The sparse

graph in [7] was generated by human editors at the All Music Guide. [7] shows that human perception of artist similarity is well modeled by generalizing using the shortest graph distance. Similar to [14], [7] projects the $N \times N$ set of artist distances into a Euclidean space by a full MDS algorithm. Note that the MDS system in [7] has $N = 412$.

The computational complexity for these methods inhibit their use on large data sets. Let us analyze the complexity for each portion of this method.

For finding all of the minimum distances, Floyd's algorithm operates on a dense matrix of distances and has computational complexity $O(N^3)$. A better choice is to run Dijkstra's algorithm [6], which finds the minimum distances from a single vertex to all other vertices in the graph. Thus, Dijkstra's algorithm must be run $N$ times. The complexity of one invocation of Dijkstra's algorithm (when implemented with a binary heap [11]) is $O(M \log N)$ where $M$ is the number of edges in the graph.

Running a standard MDS algorithm on a full $N \times N$ matrix of distances requires $O(N^2 K d)$ computation, where $K$ is the number of iterations of the MDS algorithm and $d$ is the dimensionality of the embedding. Therefore, the overall computational complexity of the approach is $O(MN \log N + N^2 K d)$, which can be prohibitive for large $N$ and $M$.

3. *Use a graph algorithm to generate a thin dense rectangle of distances.*

One natural way to reduce the complexity of the graph traversal part of Isomap is to not run Dijkstra's algorithm $N$ times. In other words, instead of generating the entire $N \times N$ matrix of dissimilarities, generate an interesting subset of $n$ rows, $n << N$.

There are a family of MDS algorithms, here called *Rectangular Dijkstra* (RD) MDS algorithms. RD algorithms operate on a dense rectangle of distances, filled in by Dijkstra's algorithm. The first published member of this family was Landmark MDS (LMDS) [5]. Bengio, et al.[2] show that LMDS is the Nyström approximation [1] combined with classical MDS [4] operating on the rectangular distance matrix. (See also [10] for Nyström applied to spectral clustering).

LMDS operates on a number of rows proportional to the embedding dimensionality, $d$. Thus, Dijkstra gets called $O(d)$ times. LMDS then centers the $n \times n$ distance submatrix, converting it into a kernel matrix $\mathbf{K}$. The top $d$ column eigenvectors ($\vec{v}_i$) and eigenvalues $\lambda_i$ of $\mathbf{K}$ are then computed. The embedding coordinate for the $m$th point is thus

$$\vec{x}_m = \frac{1}{2} \sum_j M_{ij}(A_j - D_{jm}), \tag{2}$$

where $M_{ij} = \vec{v}_i^T / \sqrt{\lambda_i}$, $A_j$ is the average distance in the $j$th row of the rectangular distance matrix and $D_{jm}$ is the distance between the $m$th point and the $j$th point ($j \in [1..n]$). Thus, the computational complexity of LMDS is $O(Md \log N + Nd^2 + d^3)$.

## 3 New Algorithm: Fast Sparse Embedding

LMDS requires the solution of an $n \times n$ eigenproblem. To avoid this eigenproblem, this paper presents a new RD MDS algorithm, called FSE (Fast Sparse Embedding). Instead of a Nyström approximation, FSE uses FastMap [8]: an MDS algorithm that takes a constant number of rows of the dissimilarity matrix. FastMap iterates over the dimensions of the projection, fixing the position of all vertices in each dimension in turn. FastMap thus approximates the solution of the eigenproblem through deflation.

Consider the first dimension. Two vertices ($\vec{x}_a, \vec{x}_b$) are chosen and the dissimilarity from these two vertices to all other vertices $i$ are computed: ($\delta_{ai}, \delta_{bi}$). In FSE, these dissimilarities are computed by Dijkstra's algorithm. During the first iteration (dimension), the distances ($d_{ai}, d_{bi}$) are set equal to the dissimilarities.

The 2*N* distances can determine the location of the vertices along the dimension up to a shift, through use of the law of cosines:

$$x_i = \frac{d_{ai}^2 - d_{bi}^2}{2d_{ab}}. \tag{3}$$

For each subsequent dimension, two new vertices are chosen and new dissimilarities $(\delta_{ai}, \delta_{bi})$ are computed by Dijkstra's algorithm. The subsequent dimensions are assumed to be orthogonal to previous ones, so the distances for dimension *N* are computed from the dissimilarities via:

$$\delta_{ai}^2 = d_{ai}^2 + \sum_{n=1}^{N-1}(x_{an} - x_{in})^2 \Rightarrow d_{ai}^2 = \delta_{ai}^2 - \sum_{n=1}^{N-1}(x_{an} - x_{in})^2. \tag{4}$$

Thus, each dimension accounts for a fraction of the dissimilarity matrix, analogous to PCA. Note that, except for $d_{ab}$, all other distances are needed as distance squared, so only one square root for each dimension is required. The distances produced by Dijkstra's algorithm are the minimum graph distances modified by equation (4) in order to reflect the projection used so far.

For each dimension, the vertices *a* and *b* are heuristically chosen to be as far apart as possible. In order to avoid an $O(N^2)$ step in choosing *a* and *b*, [8] recommends starting with an arbitrary point, finding the point furthest away from the current point, then setting the current point to the farthest point and repeating.

The work of each Dijkstra call (including equation (4)) is $O(M\log N + Nd)$, so the complexity of the entire algorithm is $O(Md\log N + Nd^2)$.

# 4 Experimental Results

## 4.1 Artificial Data

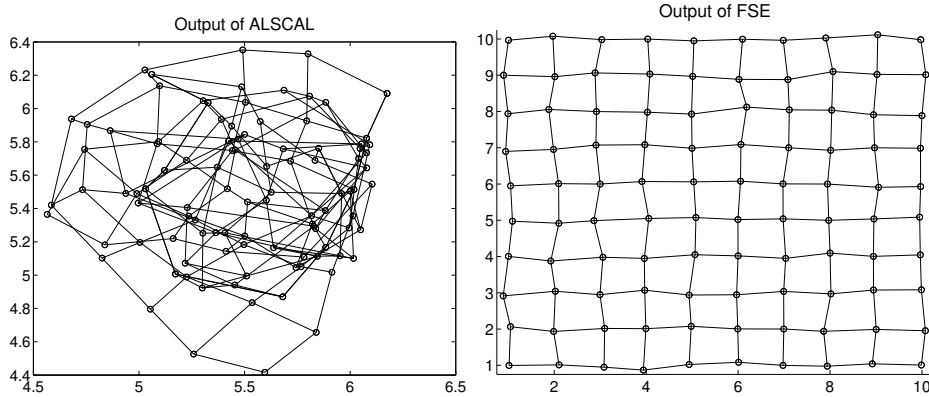

Figure 1: Reconstructing a grid of points directly from a sparse distance matrix. On the left, ALSCAL cannot reconstruct the grid, while on the right, FSE accurately reconstructs the grid.

An MDS algorithm needs to be tested on distance matrices that are computed from distances between real points, in order to verify that the algorithm quickly produces sensible results.

FSE and ALSCAL were both tested on a set of 100 points in a $10 \times 10$ 2D grid with unit spacing. The distance from each point to a random 10 of the nearest 20 other points were presented to each algorithm. The results are shown in Figure 1. Procrustes analysis [4] is applied to output of each algorithm; the output is shown after the best orthogonal affine projection between the algorithm output and the original data.

Figure 1 shows that ALSCAL does a very poor job of reconstructing the locations of the data points, while FSE accurately reconstructs the grid locations. ALSCAL's poor performance is caused by performing optimization on a non-convex cost function. When the dissimilarity matrix is very sparse, there are not enough constraints on the final solution, so ALSCAL gets stuck in a local minimum. Similar results were seen from Sammon's method [4].

These results show that FSE (and other RD MDS algorithms) are preferable to using sparse MDS algorithms. FSE does not solve an optimization problem, hence does not get stuck in a local minimum.

### 4.2 Application: Generalizing Music Similarity

This section presents the results of using RD MDS algorithms to project a large music dissimilarity graph into low-dimensional Euclidean space. This projection enables visualization and interpolation over music collections.

The dissimilarity graph was derived from a music metadata database. The database consists of 10289 artists, 67799 albums, and 188749 tracks. Each track has subjective metadata assigned to it by human editors: style (specific style), subgenre (more general style), vocal code (gender of singer), and mood. See [12] for more details on the metadata. The database contains which tracks occur on which albums and which artists created those albums.

| Relationship Between Entities | Edge Distance in Graph |
|---|---|
| Two tracks have same style, vocal code, mood | 1 |
| Two tracks have same style | 2 |
| Two tracks have same subgenre | 4 |
| Track is on album | 1 |
| Album is by artist | 2 |

Table 1: Mapping of relationship to edge distance.

A sparse similarity graph was extracted from the metadata database according to Table 1. Every track, album, and artist are represented by a vertex in the graph. Every track was connected to all albums it appeared on, while each album was connected to its artist. The track similarity edges were sampled randomly, to provide an average of 7 links of edges of distance 1, 2, and 4. The final graph contained 267K vertices and 3.22M edges. RD MDS enabled this experiment: the full distance matrix would have taken days to compute with 267K calls to Dijkstra. Also, the graph distances were derived after some tuning (not on the test set): the speed of RD MDS enabled this tuning.

One advantage of the music application is that the quality of the embedding can be tested externally. A test set of 50 playlists, with 444 pairs of sequential songs was gathered from real users who listened to these playlists. An embedding is considered good if sequential songs in the playlists are frequently closer to each other than random songs in the database. Table 2 shows the quality of the embedding as a fraction of random songs that are closer than sequential songs. The lower the fraction, the better the embedding, because the embedding more accurately reflects users' ideas of music similarity. This fraction is computed by treating the pairwise distances as scores from a classifier, computing an ROC curve, then computing 1.0-the area under the ROC curve [3].

| Algorithm | $n$ | Average % of Random Songs Closer than Sequential Songs | CPU time (sec) |
|---|---|---|---|
| FSE | 60 | 5.0% | 52.8 |
| LMDS | 60 | 4.5% | 52.7 |
| LMDS | 100 | 4.1% | 87.4 |
| LMDS | 200 | 3.3% | 175.0 |
| LMDS | 400 | 3.2% | 355.1 |
| Laplacian Eigenmaps | N/A | 13.0% | 8003.4 |

Table 2: Speed and accuracy of music embedding for various algorithms.

All embeddings are 20-dimensional ($d = 20$). The CPU time was measured on a 2.4 GHz Pentium 4. FSE uses a fixed rectangle size $n = 3d$, so has one entry in the table. For the same $n$, FSE and LMDS are competitive. However, LMDS can trade off speed for accuracy by increasing $n$.

A Laplacian Eigenmap applied to the entire sparse similarity matrix was much slower than either of the RD MDS algorithms, and did not perform as well for this problem. A Gaussian kernel with $\sigma = 2$ was used to convert distances to similarities for the Laplacian Eigenmap. The slowness of the Laplacian eigenmap prevented extensive tuning of the parameters.

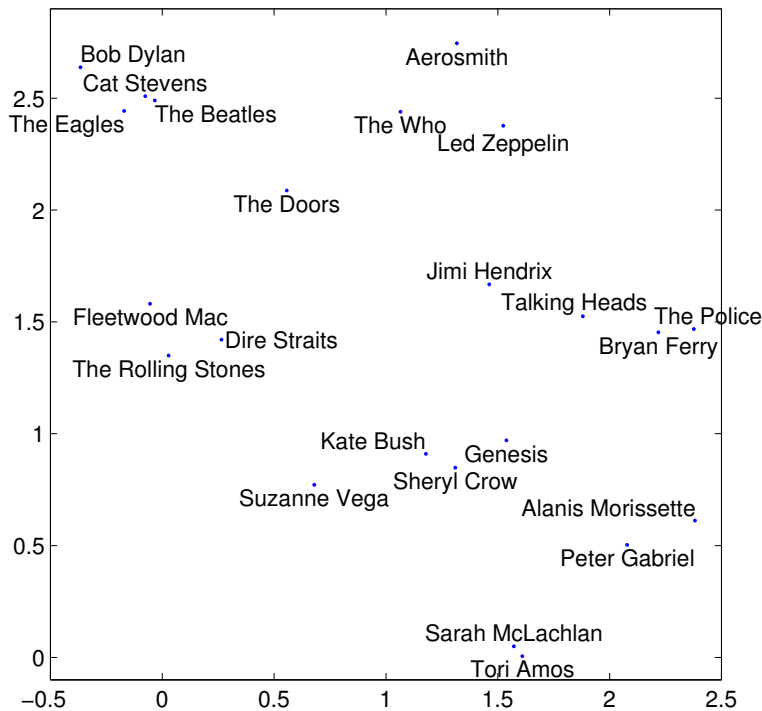

Figure 2: LMDS Projection of the entire music dissimilarity graph into 2D. The coordinates of 23 artists are shown.

Given that LMDS outperforms FSE for large $n$, this paper now presents qualitative results from the LMDS $n = 400$ projection. First, the top two dimensions are plotted to form a visualization of music space. This visualization is shown in Figure 4.2, which shows the

coordinates of 23 artists that occur near the center of the space. Even restricted to the top two dimensions, the projection is sensible. For example, Tori Amos and Sarah McLachlan are mapped to be very close.

| Artist 1 | Track 1 | Artist 2 | Track 2 |
|---|---|---|---|
| Jimi Hendrix | Purple Haze | Alanis | Hand In My Pocket |
| Jimi Hendrix | Fire | Alanis | All I Really Want |
| Jimi Hendrix | Red House | Alanis | You Oughta Know |
| Jimi Hendrix | I Don't Live Today | Alanis | Right Through You |
| Jimi Hendrix | Foxey Lady | Alanis | You Learn |
| Jimi Hendrix | 3rd Stone from the Sun | Alanis | Ironic |
| Doors | Waiting for the Sun | Sarah McLachlan | Full of Grace |
| Doors | LA Woman | Sarah McLachlan | Hold On |
| Doors | Riders on the Storm | Sarah McLachlan | Good Enough |
| Doors | Love her Madly | Sarah McLachlan | The Path of Thorns |
| Cat Stevens | Ready | Sarah McLachlan | Possession |
| Cat Stevens | Music | Blondie | Tide is High |
| Cat Stevens | Jesus | Sarah McLachlan | Ice Cream |
| Cat Stevens | King of Trees | Sarah McLachlan | Fumbling Towards Ecstasy |
| The Beatles | Octopus's Garden | Fiona Apple | Limp |
| The Beatles | I'm So Tired | Fiona Apple | Paper Bag |
| The Beatles | Revolution 9 | Fiona Apple | Fast As You Can |
| The Beatles | Sgt. Pepper's Lonely | Blondie | Call Me |
| The Beatles | Please Please Me | Blondie | Hanging on the Telephone |
| The Beatles | Eleanor Rigby | Blondie | Rapture |

Table 3: Two playlists produced by the system. Each playlist reads top to bottom. The playlists interpolate between the first and last songs.

The main application for the music graph projection is the generation of playlists. There are several different possible objectives for music playlists: background listening, dance mixes, music discovery. One of the criteria for playlists is that they play similar music together (i.e., avoid distracting jumps, like New Age to Heavy Metal). The goal for this paper is to generate playlists for background listening. Therefore, the only criterion we use for generation is smoothness and playlists are generated by linear interpolation in the embedding space.

However, smoothness is not the only possible playlist generation mode: other criteria can be added (such as matching beats or artist self-avoidance or minimum distance between songs). These criteria can be added on top of the smoothness criteria. Such criteria are a matter of subjective musical taste and are beyond the scope of this paper.

Table 3 shows two background-listening playlists formed by interpolating in the projected space. The playlists were drawn from a collection of 3920 songs. Unlike the image interpolation in [14], not every point in the 20-dimensional space has a valid song attached to it. The interpolation was performed by first computing the line segment connecting the first and last song, and then placing $K$ equally-spaced points along the line segment, where $K$ is the number of slots in the playlist. For every slot, the location of the previous song is projected onto a hyperplane normal to the line segment that goes through the $i$th point. The projected location is then moved halfway to the $i$th point, and the nearest song to the moved location is placed into the playlist. This method provides smooth interpolation without large jumps, as can be seen in Table 3.

# 5  Discussion and Conclusions

Music playlist generation and browsing can utilize a large sparse similarity graph designed by editors. In order to allow tractable computations on this graph, its vertices can be projected into a low-dimensional space. This projection enables smooth interpolation and two-dimensional display of music.

Music similarity graphs are amongst the largest graphs ever to be embedded. Rectangular Dijkstra MDS algorithms can be used to efficiently embed these large sparse graphs. This paper showed that FSE and the Nyström (LMDS) technique are both efficient and have comparable performance for the same size of rectangle. Both algorithms are much more efficient than Laplacian Eigenmaps. However, LMDS permits an accuracy/speed trade-off that makes it preferable. Using LMDS, a music graph with 267K vertices and 3.22M edges can be embedded in approximately 6 minutes.

## References

[1] C. Baker. *The numerical treatment of integral equations*. Clarendon Press, Oxford, 1977.

[2] Y. Bengio, J.-F. Paiement, and P. Vincent. Out-of-sample extensions for LLE, Isomap, MDS, Eigenmaps and spectral clustering. In S. Thrun, L. Saul, and B. Schø"lkopf, editors, *Proc. NIPS*, volume 16, 2004.

[3] A. P. Bradley. The user of area under the ROC curve in the evaluation of machine learning algorithms. *Pattern Recognition*, 30:1145–1159, 1997.

[4] T. F. Cox and M. A. A. Cox. *Multidimensional Scaling*. Number 88 in Monographs on Statistics and Applied Probability. Chapman & Hall/CRC, 2nd edition, 2001.

[5] V. de Silva and J. B. Tenenbaum. Global versus local methods in nonlinear dimensionality reduction. In S. Becker, S. Thrun, and K. Obermayer, editors, *Proc. NIPS*, volume 15, pages 721–728, 2003.

[6] E. W. Dijkstra. A note on two problems in connexion with graphs. *Numerical Mathematics*, 1:269–271, 1959.

[7] D. P. W. Ellis, B. Whitman, A. Berenzweig, and S. Lawrence. The quest for ground truth in musical artist similarity. In *Proc. International Conference on Music Information Retrieval (ISMIR)*, 2002.

[8] C. Faloutsos and K.-I. Lin. Fastmap: A fast algorithm for indexing, data-mining and visualization of traditional and multimedia databases. In *Proc. ACM SIGMOD*, pages 163–174, 1995.

[9] R. Floyd. Algorithm 97 (shortest path). *Communications of the ACM*, 7:345, 1962.

[10] C. Fowlkes, S. Belongie, and J. Malik. Efficient spatiotemporal grouping using the Nyström method. In *Proc. CVPR*, volume 1, pages I–231–I–238, 2001.

[11] D. B. Johnson. Efficient algorithms for shortest paths in sparse networks. *JACM*, 24:1–13, 1977.

[12] J. C. Platt, C. J. C. Burges, S. Swenson, C. Weare, and A. Zheng. Learning a gaussian process prior for automatically generating music playlists. In T. Dietterich, S. Becker, and Z. Ghahramani, editors, *Proc. NIPS*, volume 14, pages 1425–1432, 2002.

[13] Y. Takane, F. W. Young, and J. de Leeuw. Nonmetric individual differences multidimensional scaling: an alternating least squares method with optimal scaling features. *Psychometrika*, 42:7–67, 1977.

[14] J. B. Tenenbaum. Mapping a manifold of perceptual observations. In M. Jordan, M. Kearns, and S. Solla, editors, *Proc. NIPS*, volume 10, pages 682–688, 1998.
